# Regression with Input-Dependent Noise: A Bayesian Treatment

**Christopher M. Bishop**
C.M.Bishop@aston.ac.uk

**Cazhaow S. Qazaz**
qazazcs@aston.ac.uk

Neural Computing Research Group
Aston University, Birmingham, B4 7ET, U.K.
http://www.ncrg.aston.ac.uk/

## Abstract

In most treatments of the regression problem it is assumed that the distribution of target data can be described by a deterministic function of the inputs, together with additive Gaussian noise having constant variance. The use of maximum likelihood to train such models then corresponds to the minimization of a sum-of-squares error function. In many applications a more realistic model would allow the noise variance itself to depend on the input variables. However, the use of maximum likelihood to train such models would give highly biased results. In this paper we show how a Bayesian treatment can allow for an input-dependent variance while overcoming the bias of maximum likelihood.

## 1 Introduction

In regression problems it is important not only to predict the output variables but also to have some estimate of the error bars associated with those predictions. An important contribution to the error bars arises from the intrinsic noise on the data. In most conventional treatments of regression, it is assumed that the noise can be modelled by a Gaussian distribution with a constant variance. However, in many applications it will be more realistic to allow the noise variance itself to depend on the input variables. A general framework for modelling the conditional probability density function of the target data, given the input vector, has been introduced in the form of *mixture density networks* by Bishop (1994, 1995). This uses a feed-forward network to set the parameters of a mixture kernel distribution, following Jacobs *et al.* (1991). The special case of a single isotropic Gaussian kernel function

was discussed by Nix and Weigend (1995), and its generalization to allow for an arbitrary covariance matrix was given by Williams (1996).

These approaches, however, are all based on the use of maximum likelihood, which can lead to the noise variance being systematically under-estimated. Here we adopt an approximate hierarchical Bayesian treatment (MacKay, 1991) to find the most probable interpolant and most probable input-dependent noise variance. We compare our results with maximum likelihood and show how this Bayesian approach leads to a significantly reduced bias.

In order to gain some insight into the limitations of the maximum likelihood approach, and to see how these limitations can be overcome in a Bayesian treatment, it is useful to consider first a much simpler problem involving a single random variable (Bishop, 1995). Suppose that a variable $z$ is known to have a Gaussian distribution, but with unknown mean $\mu$ and unknown variance $\sigma^2$. Given a sample $D \equiv \{z_n\}$ drawn from that distribution, where $n = 1, \ldots, N$, our goal is to infer values for the mean and variance. The likelihood function is given by

$$p(D|\mu, \sigma^2) = \frac{1}{(2\pi\sigma^2)^{N/2}} \exp\left\{ -\frac{1}{2\sigma^2} \sum_{n=1}^{N} (z_n - \mu)^2 \right\}. \tag{1}$$

A non-Bayesian approach to finding the mean and variance is to maximize the likelihood jointly over $\mu$ and $\sigma^2$, corresponding to the intuitive idea of finding the parameter values which are most likely to have given rise to the observed data set. This yields the standard result

$$\widehat{\mu} = \frac{1}{N} \sum_{n=1}^{N} z_n, \qquad \widehat{\sigma}^2 = \frac{1}{N} \sum_{n=1}^{N} (z_n - \widehat{\mu})^2. \tag{2}$$

It is well known that the estimate $\widehat{\sigma}^2$ for the variance given in (2) is *biased* since the expectation of this estimate is not equal to the true value

$$\mathcal{E}[\widehat{\sigma}^2] = \frac{N-1}{N} \sigma_0^2 \tag{3}$$

where $\sigma_0^2$ is the true variance of the distribution which generated the data, and $\mathcal{E}[\cdot]$ denotes an average over data sets of size $N$. For large $N$ this effect is small. However, in the case of regression problems there are generally much larger number of degrees of freedom in relation to the number of available data points, in which case the effect of this bias can be very substantial.

The problem of bias can be regarded as a symptom of the maximum likelihood approach. Because the mean $\widehat{\mu}$ has been estimated from the data, it has fitted some of the noise on the data and this leads to an under-estimate of the variance. If the true mean is used in the expression for $\widehat{\sigma}^2$ in (2) instead of the maximum likelihood expression, then the estimate is unbiased.

By adopting a Bayesian viewpoint this bias can be removed. The marginal likelihood of $\sigma^2$ should be computed by *integrating* over the mean $\mu$. Assuming a 'flat' prior $p(\mu)$ we obtain

$$p(D|\sigma^2) = \int p(D|\sigma^2, \mu) p(\mu) \, d\mu \tag{4}$$

$$\propto \quad \frac{1}{\sigma^{N-1}} \exp\left\{-\frac{1}{2\sigma^2}\sum_{n=1}^{N}(z_n - \widehat{\mu})^2\right\}. \tag{5}$$

Maximizing (5) with respect to $\sigma^2$ then gives

$$\widetilde{\sigma}^2 = \frac{1}{N-1}\sum_{n=1}^{N}(z_n - \widehat{\mu})^2 \tag{6}$$

which is unbiased.

This result is illustrated in Figure 1 which shows contours of $p(D|\mu, \sigma^2)$ together with the marginal likelihood $p(D|\sigma^2)$ and the conditional likelihood $p(D|\widehat{\mu}, \sigma^2)$ evaluated at $\mu = \widehat{\mu}$.

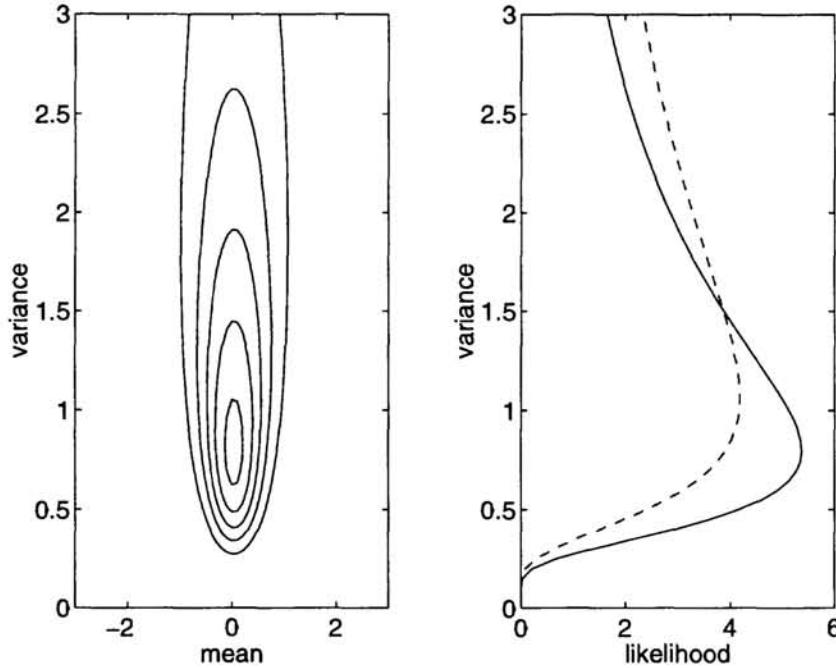

Figure 1: The left hand plot shows contours of the likelihood function $p(D|\mu, \sigma^2)$ given by (1) for 4 data points drawn from a Gaussian distribution having zero mean and unit variance. The right hand plot shows the marginal likelihood function $p(D|\sigma^2)$ (dashed curve) and the conditional likelihood function $p(D|\widehat{\mu}, \sigma^2)$ (solid curve). It can be seen that the skewed contours result in a value of $\widehat{\sigma}^2$, which maximizes $p(D|\widehat{\mu}, \sigma^2)$, which is smaller than $\widetilde{\sigma}^2$ which maximizes $p(D|\sigma^2)$.

## 2   Bayesian Regression

Consider a regression problem involving the prediction of a noisy variable $t$ given the value of a vector $\mathbf{x}$ of input variables[1]. Our goal is to predict both a regression function and an input-dependent noise variance. We shall therefore consider two networks. The first network takes the input vector $\mathbf{x}$ and generates an output

$y(\mathbf{x}; \mathbf{w})$ which represents the regression function, and is governed by a vector of weight parameters $\mathbf{w}$. The second network also takes the input vector $\mathbf{x}$, and generates an output function $\beta(\mathbf{x}; \mathbf{u})$ representing the inverse variance of the noise distribution, and is governed by a vector of weight parameters $\mathbf{u}$. The conditional distribution of target data, given the input vector, is then modelled by a normal distribution $p(t|\mathbf{x}, \mathbf{w}, \mathbf{u}) = \mathcal{N}(t|y, \beta^{-1})$. From this we obtain the likelihood function

$$p(D|\mathbf{w}, \mathbf{u}) = \frac{1}{Z_D} \exp\left\{-\sum_{n=1}^{N} \beta_n E_n\right\} \tag{7}$$

where $\beta_n = \beta(\mathbf{x}_n; \mathbf{u})$,

$$Z_D = \prod_{n=1}^{N} \left(\frac{2\pi}{\beta_n}\right)^{1/2}, \qquad E_n = \frac{1}{2}(y(\mathbf{x}_n; \mathbf{w}) - t_n)^2 \tag{8}$$

and $D \equiv \{\mathbf{x}_n, t_n\}$ is the data set.

Some simplification of the subsequent analysis is obtained by taking the regression function, and $\ln \beta$, to be given by linear combinations of fixed basis functions, as in MacKay (1995), so that

$$y(\mathbf{x}; \mathbf{w}) = \mathbf{w}^{\mathrm{T}} \boldsymbol{\phi}(\mathbf{x}), \qquad \beta(\mathbf{x}; \mathbf{u}) = \exp\left(\mathbf{u}^{\mathrm{T}} \boldsymbol{\psi}(\mathbf{x})\right) \tag{9}$$

where choose one basis function in each network to be a constant $\phi_0 = \psi_0 = 1$ so that the corresponding weights $w_0$ and $u_0$ represent bias parameters.

The maximum likelihood procedure chooses values $\hat{\mathbf{w}}$ and $\hat{\mathbf{u}}$ by finding a joint maximum over $\mathbf{w}$ and $\mathbf{u}$. As we have already indicated, this will give a biased result since the regression function inevitably fits part of the noise on the data, leading to an over-estimate of $\beta(\mathbf{x})$. In extreme cases, where the regression curve passes exactly through a data point, the corresponding estimate of $\beta$ can go to infinity, corresponding to an estimated noise variance of zero.

The solution to this problem has already been indicated in Section 1 and was first suggested in this context by MacKay (1991, Chapter 6). In order to obtain an unbiased estimate of $\beta(\mathbf{x})$ we must find the marginal distribution of $\beta$, or equivalently of $\mathbf{u}$, in which we have integrated out the dependence on $\mathbf{w}$. This leads to a hierarchical Bayesian analysis.

We begin by defining priors over the parameters $\mathbf{w}$ and $\mathbf{u}$. Here we consider isotropic Gaussian priors of the form

$$p(\mathbf{w}|\alpha_w) = \left(\frac{\alpha_w}{2\pi}\right)^{1/2} \exp\left\{-\frac{\alpha_w}{2}\|\mathbf{w}\|^2\right\} \tag{10}$$

$$p(\mathbf{u}|\alpha_u) = \left(\frac{\alpha_u}{2\pi}\right)^{1/2} \exp\left\{-\frac{\alpha_u}{2}\|\mathbf{u}\|^2\right\} \tag{11}$$

where $\alpha_w$ and $\alpha_u$ are *hyper-parameters*. At the first stage of the hierarchy, we assume that $\mathbf{u}$ is fixed to its most probable value $\mathbf{u}_{\mathrm{MP}}$, which will be determined shortly. The most probable value of $\mathbf{w}$, denoted by $\mathbf{w}_{\mathrm{MP}}$, is then found by maxi-

mizing the posterior distribution[2]

$$p(\mathbf{w}|D, \mathbf{u}_{\text{MP}}, \alpha_w) = \frac{p(D|\mathbf{w}, \mathbf{u}_{\text{MP}})p(\mathbf{w}|\alpha_w)}{p(D|\mathbf{u}_{\text{MP}}, \alpha_w)} \tag{12}$$

where the denominator in (12) is given by

$$p(D|\mathbf{u}_{\text{MP}}, \alpha_w) = \int p(D|\mathbf{w}, \mathbf{u}_{\text{MP}})p(\mathbf{w}|\alpha_w)\, d\mathbf{w}. \tag{13}$$

Taking the negative log of (12), and dropping constant terms, we see that $\mathbf{w}_{\text{MP}}$ is obtained by minimizing

$$S(\mathbf{w}) = \sum_{n=1}^{N} \beta_n E_n + \frac{\alpha_w}{2}\|\mathbf{w}\|^2 \tag{14}$$

where we have used (7) and (10). For the particular choice of model (9) this minimization represents a linear problem which is easily solved (for a given $\mathbf{u}$) by standard matrix techniques.

At the next level of the hierarchy, we find $\mathbf{u}_{\text{MP}}$ by maximizing the marginal posterior distribution

$$p(\mathbf{u}|D, \alpha_u, \alpha_w) = \frac{p(D|\mathbf{u}, \alpha_w)p(\mathbf{u}|\alpha_u)}{p(D|\alpha_w, \alpha_u)}. \tag{15}$$

The term $p(D|\mathbf{u}, \alpha_w)$ is just the denominator from (12) and is found by integrating over $\mathbf{w}$ as in (13). For the model (9) and prior (10) this integral is Gaussian and can be performed analytically without approximation. Again taking logarithms and discarding constants, we have to minimize

$$M(\mathbf{u}) = \sum_{n=1}^{N} \beta_n E_n + \frac{\alpha_u}{2}\|\mathbf{u}\|^2 - \frac{1}{2}\sum_{n=1}^{N} \ln \beta_n + \frac{1}{2}\ln|\mathbf{A}| \tag{16}$$

where $|\mathbf{A}|$ denotes the determinant of the Hessian matrix $\mathbf{A}$ given by

$$\mathbf{A} = \sum_{n=1}^{N} \beta_n \phi(\mathbf{x}_n)\phi(\mathbf{x}_n)^{\text{T}} + \alpha_w \mathbf{I} \tag{17}$$

and $\mathbf{I}$ is the unit matrix. The function $M(\mathbf{u})$ in (16) can be minimized using standard non-linear optimization algorithms. We use scaled conjugate gradients, in which the necessary derivatives of $\ln|\mathbf{A}|$ are easily found in terms of the eigenvalues of $\mathbf{A}$.

In summary, the algorithm requires an outer loop in which the most probable value $\mathbf{u}_{\text{MP}}$ is found by non-linear minimization of (16), using the scaled conjugate gradient algorithm. Each time the optimization code requires a value for $M(\mathbf{u})$ or its gradient, for a new value of $\mathbf{u}$, the optimum value for $\mathbf{w}_{\text{MP}}$ must be found by minimizing (14). In effect, $\mathbf{w}$ is evolving on a fast time-scale, and $\mathbf{u}$ on a slow time-scale. The corresponding maximum (penalized) likelihood approach consists of a joint non-linear optimization over $\mathbf{u}$ and $\mathbf{w}$ of the posterior distribution $p(\mathbf{w}, \mathbf{u}|D)$ obtained from (7), (10) and (11). Finally, the hyperparameters are given fixed values $\alpha_w = \alpha_u = 0.1$ as this allows the maximum likelihood and Bayesian approaches to be treated on an equal footing.

## 3  Results and Discussion

As an illustration of this algorithm, we consider a toy problem involving one input and one output, with a noise variance which has an $x^2$ dependence on the input variable. Since the estimated quantities are noisy, due to the finite data set, we consider an averaging procedure as follows. We generate 100 independent data sets each consisting of 10 data points. The model is trained on each of the data sets in turn and then tested on the remaining 99 data sets. Both the $y(\mathbf{x}; \mathbf{w})$ and $\beta(\mathbf{x}; \mathbf{u})$ networks have 4 Gaussian basis functions (plus a bias) with width parameters chosen to equal the spacing of the centres.

Results are shown in Figure 2. It is clear that the maximum likelihood results are biased and that the noise variance is systematically underestimated. By contrast,

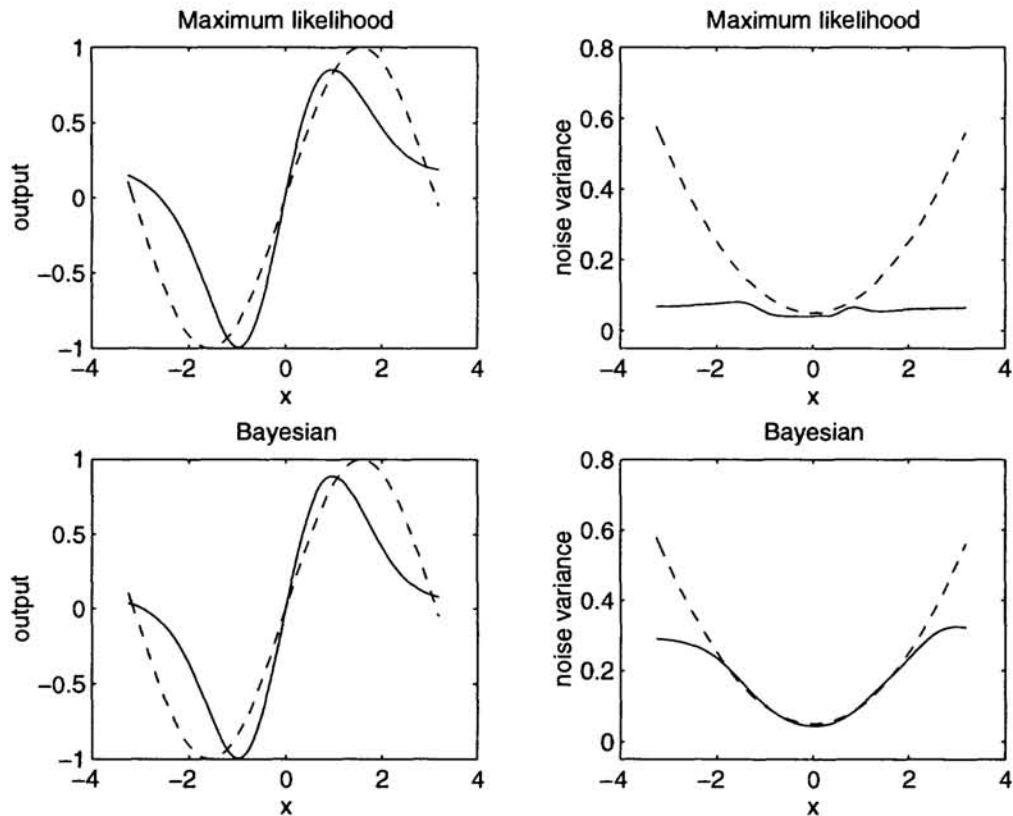

Figure 2: The left hand plots show the sinusoidal function (dashed curve) from which the data were generated, together with the regression function averaged over 100 training sets. The right hand plots show the true noise variance (dashed curve) together with the estimated noise variance, again averaged over 100 data sets.

the Bayesian results show an improved estimate of the noise variance. This is borne out by evaluating the log likelihood for the test data under the corresponding predictive distributions. The Bayesian approach gives a log likelihood per data point, averaged over the 100 runs, of $-1.38$. Due to the over-fitting problem, maximum likelihood occasionally gives extremely large negative values for the log likelihood (when $\beta$ has been estimated to be very large, corresponding to a regression curve which passes close to an individual data point). Even omitting these extreme values, the maximum likelihood still gives an average log likelihood per data point of

$-17.1$ which is substantially smaller than the Bayesian result.

We are currently exploring the use of Markov chain Monte Carlo methods (Neal, 1993) to perform the integrations required by the Bayesian analysis numerically, without the need to introduce the Gaussian approximation or the evidence framework. Recently, MacKay (1995) has proposed an alternative technique based on Gibbs sampling. It will be interesting to compare these various approaches.

**Acknowledgements:** This work was supported by EPSRC grant GR/K51792, *Validation and Verification of Neural Network Systems.*

## Footnotes

[1]For simplicity we consider a single output variable. The extension of this work to multiple outputs is straightforward.

[2]Note that the result will be dependent on the choice of parametrization since the maximum of a distribution is not invariant under a change of variable.

# References

Bishop, C. M. (1994). Mixture density networks. Technical Report NCRG/94/001, Neural Computing Research Group, Aston University, Birmingham, UK.

Bishop, C. M. (1995). *Neural Networks for Pattern Recognition.* Oxford University Press.

Jacobs, R. A., M. I. Jordan, S. J. Nowlan, and G. E. Hinton (1991). Adaptive mixtures of local experts. *Neural Computation* **3** (1), 79–87.

MacKay, D. J. C. (1991). *Bayesian Methods for Adaptive Models.* Ph.D. thesis, California Institute of Technology.

MacKay, D. J. C. (1995). Probabilistic networks: new models and new methods. In F. Fogelman-Soulié and P. Gallinari (Eds.), *Proceedings ICANN'95 International Conference on Artificial Neural Networks*, pp. 331–337. Paris: EC2 & Cie.

Neal, R. M. (1993). Probabilistic inference using Markov chain Monte Carlo methods. Technical Report CRG-TR-93-1, Department of Computer Science, University of Toronto, Cananda.

Nix, A. D. and A. S. Weigend (1995). Learning local error bars for nonlinear regression. In G. Tesauro, D. S. Touretzky, and T. K. Leen (Eds.), *Advances in Neural Information Processing Systems*, Volume 7, pp. 489–496. Cambridge, MA: MIT Press.

Williams, P. M. (1996). Using neural networks to model conditional multivariate densities. *Neural Computation* **8** (4), 843–854.